# Sequential noise compensation by sequential Monte Carlo method

**Kaisheng Yao and Satoshi Nakamura**
ATR Spoken Language Translation Research Laboratories
2-2-2, Hikaridai Seika-cho, Souraku-gun, Kyoto, 619-0288, Japan
E-mail: {kaisheng.yao, satoshi.nakamura}@slt.atr.co.jp

## Abstract

We present a sequential Monte Carlo method applied to additive noise compensation for robust speech recognition in time-varying noise. The method generates a set of samples according to the prior distribution given by clean speech models and noise prior evolved from previous estimation. An explicit model representing noise effects on speech features is used, so that an extended Kalman filter is constructed for each sample, generating the updated continuous state estimate as the estimation of the noise parameter, and prediction likelihood for weighting each sample. Minimum mean square error (MMSE) inference of the time-varying noise parameter is carried out over these samples by fusion the estimation of samples according to their weights. A residual resampling selection step and a Metropolis-Hastings smoothing step are used to improve calculation efficiency. Experiments were conducted on speech recognition in simulated non-stationary noises, where noise power changed artificially, and highly non-stationary Machinegun noise. In all the experiments carried out, we observed that the method can have significant recognition performance improvement, over that achieved by noise compensation with stationary noise assumption.

## 1   Introduction

Speech recognition in noise has been considered to be essential for its real applications. There have been active research efforts in this area. Among many approaches, model-based approach assumes explicit models representing noise effects on speech features. In this approach, most researches are focused on stationary or slow-varying noise conditions. In this situation, environment noise parameters are often estimated before speech recognition from a small set of environment adaptation data. The estimated environment noise parameters are then used to compensate noise effects in the feature or model space for recognition of noisy speech.

However, it is well-known that noise statistics may vary during recognition. In this situation, the noise parameters estimated prior to speech recognition of the utterances is possibly not relevant to the subsequent frames of input speech if environment changes.

A number of techniques have been proposed to compensate time-varying noise effects. They can be categorized into two approaches. In the first approach, time-varying environment sources are modeled by Hidden Markov Models (HMM) or Gaussian mixtures that were trained by prior measurement of environments, so that noise compensation is a task of identification of the underlying state sequences of the noise HMMs, e.g., in [1], by maximum a posterior (MAP) decision. This approach requires making a model representing different conditions of environments (signal-to-noise ratio, types of noise, etc.), so that statistics at some states or mixtures obtained before speech recognition are close to the real testing environments. In the second approach, environment model parameters are assumed to be time-varying, so it is not only an inference problem but also related to environment statistics estimation during speech recognition. The parameters can be estimated by Maximum Likelihood estimation, e.g., sequential EM algorithm [2][3][4]. They can also be estimated by Bayesian methods. In the Bayesian methods, all relevant information on the set of environment parameters and speech parameters, which are denoted as $\Theta(t)$ at frame $t$, is included in the posterior distribution given observation sequence $Y(0:t)$, i.e., $p(\Theta(t)|Y(0:t))$. Except for a few cases including linear Gaussian state space model (Kalman filter), it is formidable to evaluate the distribution updating analytically. Approximation techniques are required. For example, in [5], a Laplace transform is used to approximate the joint distribution of speech and noise parameters by vector Taylor series. The approximated joint distribution can give analytical formula for posterior distribution updating.

We report an alternative approach for Bayesian estimation and compensation of noise effects on speech features. The method is based on sequential Monte Carlo method [6]. In the method, a set of samples is generated hierarchically from the prior distribution given by speech models. A state space model representing noise effects on speech features is used explicitly, and an extended Kalman filter (EKF) is constructed in each sample. The prediction likelihood of the EKF in each sample gives its weight for selection, smoothing, and inference of the time-varying noise parameter, so that noise compensation is carried out afterwards. Since noise parameter estimation, noise compensation and speech recognition are carried out frame-by-frame, we denote this approach as sequential noise compensation.

## 2  Speech and noise model

Our work is on speech features derived from Mel Frequency Cepstral Coefficients (MFCC). It is generated by transforming signal power into log-spectral domain, and finally, by discrete Cosine transform (DCT) to the cepstral domain. The following derivation of the algorithm is in log-spectral domain. Let $t$ denote frame (time) index.

In our work, speech and noise are respectively modeled by HMMs and a Gaussian mixture. For speech recognition in stationary additive noise, the following formula [4] has been shown to be effective in compensating noise effects. For Gaussian mixture $k_t$ at state $s_t$, the Log-Add method transforms the mean vector $\mu^l_{s_t k_t}$ of the Gaussian mixture by,

$$\hat{\mu}^l_{s_t k_t} = \mu^l_{s_t k_t} + \log(1 + \exp(\mu^l_n - \mu^l_{s_t k_t})) \tag{1}$$

where $\mu^l_n$ is the mean vector in the noise model. $s_t \in \{1, \cdots, S\}, k_t \in \{1, \cdots, M\}$. $S$ and $M$ each denote the number of states in speech models and the number of mixtures at each state. Superscript $l$ indicates that parameters are in the log-spectral domain.

After the transformation, the mean vector $\hat{\mu}^l_{s_t k_t}$ is further transformed by DCT,

and then plugged into speech models for recognition of noisy speech. In case of time-varying noise, the $\mu_n^l$ should be a function of time, i.e., $\mu_n^l(t)$. Accordingly, the compensated mean is $\hat{\mu}_{s_t k_t}^l(t)$.

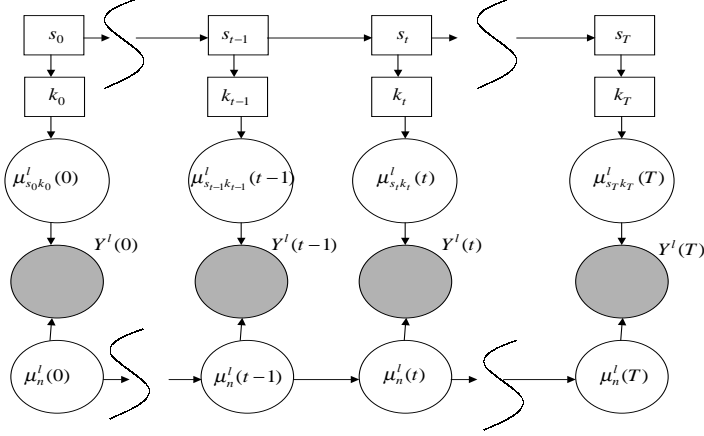

Figure 1: The graphical model representation of the dependences of the speech and noise model parameters. $s_t$ and $k_t$ each denote the state and Gaussian mixture at frame $t$ in speech models. $\mu_{s_t k_t}^l(t)$ and $\mu_n^l(t)$ each denote the speech and noise parameter. $Y^l(t)$ is the noisy speech observation.

The following analysis can be viewed in Figure 1. In Gaussian mixture $k_t$ at state $s_t$ of speech model, speech parameter $\mu_{s_t k_t}^l(t)$ is assumed to be distributed in Gaussian with mean $\mu_{s_t k_t}^l$ and variance $\Sigma_{s_t k_t}^l$. On the other hand, since the environment parameter is assumed to be time varying, the evolution of the environment mean vector can be modeled by a random walk function, i.e.,

$$\mu_n^l(t) = \mu_n^l(t-1) + v(t) \tag{2}$$

where $v(t)$ is the environment driving noise in Gaussian distribution with zero mean and variance $V$.

Then, we have,

$$p(s_t, k_t, \mu_{s_t k_t}^l(t), \mu_n^l(t) | s_{t-1}, k_{t-1}, \mu_{s_{t-1} k_{t-1}}^l(t-1), \mu_n^l(t-1))$$
$$= a_{s_{t-1} s_t} p_{s_t k_t} N(\mu_{s_t k_t}^l(t); \mu_{s_t k_t}^l, \Sigma_{s_t k_t}^l) N(\mu_n^l(t); \mu_n^l(t-1), V) \tag{3}$$

where $a_{s_{t-1} s_t}$ is the state transition probability from $s_{t-1}$ to $s_t$, and $p_{s_t k_t}$ is the mixture weight. The above formula gives the prior distribution of the set of speech and noise model parameter $\Theta(t) = \{s_t, k_t, \mu_{s_t k_t}^l(t), \mu_n^l(t)\}$.

Furthermore, given observation $Y^l(t)$, assume that the transformation by (1) has modeling and measurement uncertainty in Gaussian distribution, i.e.,

$$Y^l(t) = \mu_{s_t k_t}^l(t) + \log\left(1 + \exp\left(\mu_n^l(t) - \mu_{s_t k_t}^l(t)\right)\right) + w_{s_t k_t}(t) \tag{4}$$

where $w_{s_t k_t}(t)$ is Gaussian with zero mean and variance $\Sigma_{s_t k_t}^l$, i.e., $N(\cdot; 0, \Sigma_{s_t k_t}^l)$. Thus, the likelihood of observation $Y^l(t)$ at state $s_t$ and mixture $k_t$ is

$$p(Y^l(t)|\Theta(t)) = N(Y^l(t); \mu_{s_t k_t}^l(t) + \log\left(1 + \exp\left(\mu_n^l(t) - \mu_{s_t k_t}^l(t)\right)\right), \Sigma_{s_t k_t}^l) \tag{5}$$

Refereeing to (3) and (5), the posterior distribution of $\Theta(t)$ given $Y^l(t)$ is

$$p(s_t, k_t, \mu_{s_t k_t}^l(t), \mu_n^l(t)|Y^l(t)) \propto$$
$$p(Y^l(t)|\Theta(t))a_{s_{t-1}s_t}p_{s_t k_t}N(\mu_{s_t k_t}^l(t); \mu_{s_t k_t}^l, \Sigma_{s_t k_t}^l)N(\mu_n^l(t); \mu_n^l(t-1), V) \quad (6)$$

The time-varying noise parameter is estimated by MMSE, given as,

$$\hat{\mu}_n^l(t) = \int_{\mu_n^l(t)} \mu_n^l(t) \sum_{s_t, k_t} \int_{\mu_{s_t k_t}^l(t)} p(\Theta(t)|Y^l(0:t))d\mu_{s_t k_t}^l(t)d\mu_n^l(t) \quad (7)$$

However, it is difficult to obtain the posterior distribution $p(\Theta(t)|Y^l(0:t))$ analytically, since $p(\mu_{s_t k_t}^l(t), \mu_n^l(t)|Y^l(t))$ is non-Gaussian in $\mu_{s_t k_t}^l(t)$ and $\mu_n^l(t)$ due to the non-linearity in (4). It is thus difficult, if possible, to assign conjugate prior of $\mu_n^l(t)$ to the likelihood function $p(Y^l(t)|\Theta(t))$. Another difficulty is that the speech state and mixture sequence is hidden in (7). We thus rely on the solution by computational Bayesian approach [6].

## 3 Time-varying noise parameter estimation by sequential Monte Carlo method

We apply the sequential Monte Carlo method [6] for posterior distribution updating. At each frame $t$, a proposal importance distribution is sampled whose target is the posterior distribution in (7), and it is implemented by sampling from lower distributions in hierarchy. The method goes through the sampling, selection, and smoothing steps frame-by-frame. MMSE inference of the time-varying noise parameter is a by-product of the steps, carried out after the smoothing step.

In the sampling step, the prior distribution given by speech models is set to the proposal importance distribution, i.e., $q(\Theta(t)|\Theta(t-1)) = a_{s_{t-1}s_t}p_{s_t k_t}N(\mu_{s_t k_t}^l(t); \mu_{s_t k_t}^l, \Sigma_{s_t k_t}^l)$. The samples are then generated by sampling hierarchically of the prior distribution described as follows: set $i=1$ and perform the following steps:

1. sample $s_t^{(i)} \sim a_{s_{t-1}^{(i)}s_t}$

2. sample $k_t^{(i)} \sim p_{s_t^{(i)}k_t}$

3. sample $\mu_{s_t^{(i)}k_t^{(i)}}^{l(i)}(t) \sim N(; \mu_{s_t^{(i)}k_t^{(i)}}^l, \Sigma_{s_t^{(i)}k_t^{(i)}}^l)$, and set $i=i+1$

4. repeat step 1 to 3 until $i=N$

where superscript $(i)$ denotes the index of samples and $N$ denotes the number of samples. Each sample represents certain speech and noise parameter, which is denoted as $\Theta^{(i)}(t) = (s_t^{(i)}, k_t^{(i)}, \mu_{s_t^{(i)}k_t^{(i)}}^{l(i)}(t), \mu_n^{l(i)}(t))$. The weight of each sample is given by $\prod_{\tau=1}^t \frac{p(\Theta(\tau)^{(i)}|Y^l(\tau))}{q(\Theta(\tau)^{(i)}|\Theta(\tau-1)^{(i)})}$. Refereeing to (6), the weight is calculated by

$$\beta^{(i)}(t) = p(Y^l(t)|\Theta^{(i)}(t))N(\mu_n^{l(i)}(t); \mu_n^{l(i)}(t-1), V)\breve{\beta}^{(i)}(t-1) \quad (8)$$

where $\breve{\beta}^{(i)}(t-1)$ is the sample weight from previous frame. The remaining part in the right side of above equation, in fact, represents the prediction likelihood of the state space model given by (2) and (4) for each sample $(i)$. This likelihood can be obtained analytically since after linearization of (4) with respect to $\mu_n^l(t)$ at

$\mu_n^{l(i)}(t-1)$, an extended Kalman filter (EKF) can be obtained, where the prediction likelihood of the EKF gives the weight, and the updated continuous state of EKF gives $\mu_n^{l(i)}(t)$.

In practice, after the above sampling step, the weights of all but several samples may become insignificant. Given the fixed number of samples, this will results in degeneracy of the estimation, where not only some computational resources are wasted, but also estimation might be biased because of losing detailed information on some parts important to the parameter estimation. A selection step by residual resampling [6] is adopted after the sampling step. The method avoids the degeneracy by discarding those samples with insignificant weights, and in order to keep the number of the samples constant, samples with significant weights are duplicated. Accordingly, the weights after the selection step are also proportionally redistributed. Denote the set of samples after the selection step as $\tilde{\Theta}(t) = \{\tilde{\Theta}^{(i)}(t); i = 1 \cdots N\}$ with weights $\tilde{\beta}(t) = \{\tilde{\beta}^{(i)}(t); i = 1 \cdots N\}$.

After the selection step at frame $t$, these $N$ samples are distributed approximately according to the posterior distribution in (7). However, the discrete nature of the approximation can lead to a skewed importance weights distribution, where the extreme case is all the samples have the same $\tilde{\Theta}(t)$ estimated. A Metropolis-Hastings smoothing [7] step is introduced in each sample where the step involves sampling a candidate $\Theta^{\star(i)}(t)$ given the current $\tilde{\Theta}^{(i)}(t)$ according to the proposal importance distribution $q(\Theta^{\star}(t)|\tilde{\Theta}^{(i)}(t))$. The Markov chain then moves towards $\Theta^{\star(i)}(t)$ with acceptance possibility as $\min\{1, \frac{p(\Theta^{\star(i)}|Y^l(t))q(\tilde{\Theta}^{(i)}|\Theta^{\star(i)})}{p(\tilde{\Theta}^{(i)}|Y^l(t))q(\Theta^{\star(i)}|\tilde{\Theta}^{(i)})}\}$, otherwise it remains at $\tilde{\Theta}^{(i)}$. To simplify calculation, we assume that the importance distribution $q(\Theta^{\star}(t)|\tilde{\Theta}^{(i)}(t))$ is symmetric, and after some mathematical manipulation, it is shown that the acceptance possibility is given by $\min\{1, \frac{\beta^{\star(i)}(t)}{\tilde{\beta}^{(i)}(t)}\}$. Denote the obtained samples as $\check{\Theta}(t) = \{\check{\Theta}^{(i)}(t); i = 1 \cdots N\}$ with weights $\check{\beta}(t) = \{\check{\beta}^{(i)}(t); i = 1 \cdots N\}$.

Noise parameter $\mu_n^l(t)$ is estimated via MMSE over the samples, i.e.,

$$\hat{\mu}_n^l(t) = \sum_{i=1}^{N} \frac{\check{\beta}^{(i)}(t)}{\sum_{j=1}^{N} \check{\beta}^{(j)}(t)} \check{\mu}_n^{l(i)}(t)$$

where $\check{\mu}_n^{l(i)}(t)$ is the updated continuous state of the EKF in the sample after the smoothing step. Once the estimate $\hat{\mu}_n^l(t)$ has been obtained, it is plugged into (1) to do non-linear transformation of clean speech models.

## 4 Experimental results

### 4.1 Experimental setup

Experiments were performed on the TI-Digits database down-sampled to 16kHz. Five hundred clean speech utterances from 15 speakers and 111 utterances unseen in the training set were used for training and testing, respectively. Digits and silence were respectively modeled by 10-state and 3-state whole word HMMs with 4 diagonal Gaussian mixtures in each state.

The window size was 25.0ms with a 10.0ms shift. Twenty-six filter banks were used in the binning stage. The features were MFCC + $\Delta$ MFCC. The baseline system had a 98.7% Word Accuracy under clean conditions.

We compared three systems. The first was the baseline trained on clean speech without noise compensation, and the second was the system with noise compensation by (1) assuming stationary noise [4]. They were each denoted as Baseline and Stationary Compensation. The sequential method was un-supervised, i.e., without training transcript, and it was denoted according to the number of samples and variance of the environment driving noise $V$. Four seconds of contaminating noise was used in each experiment to obtain noise mean vector $\mu_n^l$ in (1) for Stationary Compensation. It was also for initialization of $\mu_n^l(0)$ in the sequential method. The initial $\mu_n^{l(i)}(0)$ for each sample was sampled from $N(\mu_n^l(0), 0.01) + N(\mu_n^l(0) + \zeta(0), 10.0)$, where $\zeta(0)$ was flat distribution in $[-1.0, 9.0]$.

## 4.2 Speech recognition in simulated non-stationary noise

White noise signal was multiplied by a Chirp signal and a rectangular signal, so that the noise power of the contaminating White noise changed continuously, denoted as experiment A, and dramatically, denoted as experiment B. As a result, signal-to-noise ratio (SNR) of the contaminating noise ranged from 0dB to 20.4dB. We plotted the noise power in 12th filter bank versus frames in Figure 2, together with the estimated noise power by the sequential method with number of samples set to 120 and environment driving noise variance set to 0.0001. As a comparison, we also plotted the noise power and its estimate by the method with the same number of samples but larger driving noise variance to 0.001.

By Figure 2 and Figure 3, we have the following observations. First, the method can track the evolution of the noise power. Second, the larger driving noise variance $V$ will make faster convergence but larger estimation error of the method. In terms of recognition performance, Table 1 shows that the method can effectively improve system robustness to the time-varying noise. For example, with 60 samples, and the environment driving noise variance $V$ set to 0.001, the method can improve word accuracy from 75.30% achieved by "Stationary Compensation", to 94.28% in experiment A. The table also shows that, the word accuracies can be improved by increasing number of samples. For example, given environment driving noise variance $V$ set to 0.0001, increasing number of samples from 60 to 120, can improve word accuracy from 77.11% to 85.84% in experiment B.

Table 1: Word Accuracy (in %) in simulated non-stationary noises, achieved by the sequential Monte Carlo method in comparison with baseline without noise compensation, denoted as Baseline, and noise compensation assuming stationary noise, denoted as Stationary Compensation.

| Experiment | Baseline | Stationary Compensation | # samples = 60 | | # samples = 120 | |
|---|---|---|---|---|---|---|
| | | | $V$ | | $V$ | |
| | | | 0.001 | 0.0001 | 0.001 | 0.0001 |
| A | 48.19 | 75.30 | 94.28 | 93.98 | 94.28 | 94.58 |
| B | 53.01 | 78.01 | 82.23 | 77.11 | 85.84 | 85.84 |

## 4.3 Speech recognition in real noise

In this experiment, speech signals were contaminated by highly non-stationary Machinegun noise in different SNRs. The number of samples was set to 120, and the environment driving noise variance $V$ was set to 0.0001. Recognition performances are shown in Table 2, together with "Baseline" and "Stationary Compensation".

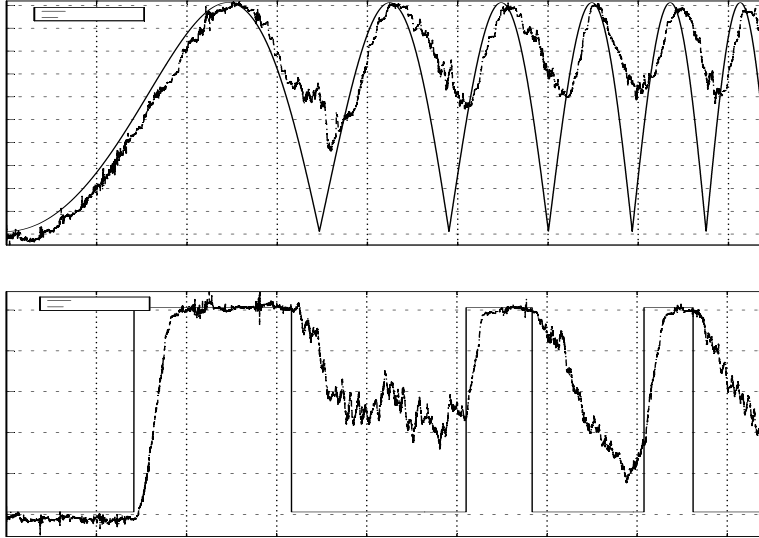

Figure 2: Estimation of the time-varying parameter $\mu_n^l(t)$ by the sequential Monte Carlo method at 12th filter bank in experiment A. Number of samples is 120. Environment driving noise variance is 0.0001. Solid curve is the true noise power. Dash-dotted curve is the estimated noise power.

It is observed that, in all SNR conditions, the method can further improve system performance, compared to that obtained by "Stationary Compensation", over "Baseline". For example, in 8.86dB SNR, the method can improve word accuracy from 75.60% by "Stationary Compensation" to 83.13%. As a whole, the method can have a relative 39.9% word error rate reduction compared to "Stationary Compensation".

Table 2: Word Accuracy (in %) in Machinegun noise, achieved by the sequential Monte Carlo method in comparison with baseline without noise compensation, denoted as Baseline, and noise compensation assuming stationary noise, denoted as Stationary Compensation.

| SNR (dB) | Baseline | Stationary Compensation | #samples = 120, $V$ = 0.0001 |
|---|---|---|---|
| 28.86 | 90.36 | 92.77 | 97.59 |
| 14.88 | 64.46 | 76.81 | 88.25 |
| 8.86 | 56.02 | 75.60 | 83.13 |
| 1.63 | 50.0 | 68.98 | 72.89 |

## 5   Summary

We have presented a sequential Monte Carlo method for Bayesian estimation of time-varying noise parameter, which is for sequential noise compensation applied to robust speech recognition. The method uses samples to approximate the posterior distribution of the additive noise and speech parameters given observation sequence.

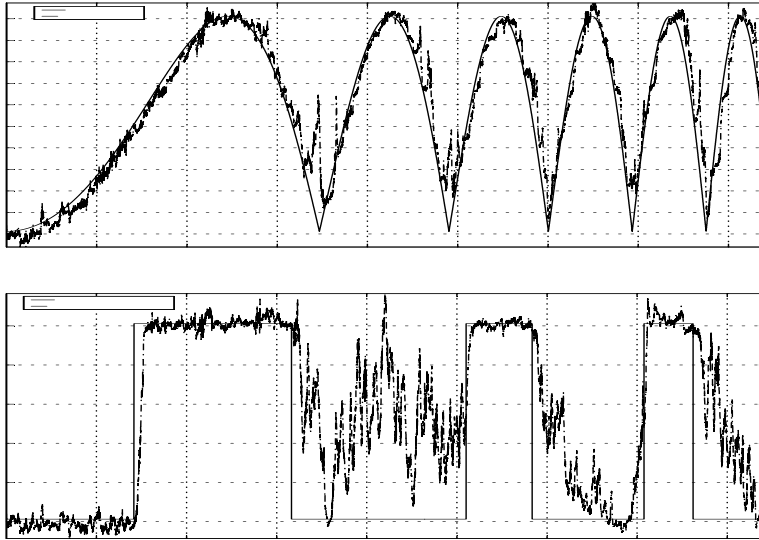

Figure 3: Estimation of the time-varying parameter $\mu_n^l(t)$ by the sequential Monte Carlo method at 12th filter bank in experiment A. Number of samples is 120. Environment driving noise variance is 0.001. Solid curve is the true noise power. Dash-dotted curve is the estimated noise power.

Once the noise parameter has been inferred, it is plugged into a non-linear transformation of clean speech models. Experiments conducted on digits recognition in simulated non-stationary noises and real noises have shown that the method is very effective to improve system robustness to time-varying additive noise.

## References

[1] A. Varga and R.K. Moore, "Hidden markov model decomposition of speech and noise," in *ICASSP*, 1990, pp. 845–848.

[2] N.S. Kim, "Nonstationary environment compensation based on sequential estimation," *IEEE Signal Processing Letters*, vol. 5, no. 3, March 1998.

[3] K. Yao, K. K. Paliwal, and S. Nakamura, "Sequential noise compensation by a sequential kullback proximal algorithm," in *EUROSPEECH*, 2001, pp. 1139–1142, extended paper submitted for publication.

[4] K. Yao, B. E. Shi, S. Nakamura, and Z. Cao, "Residual noise compensation by a sequential em algorithm for robust speech recognition in nonstationary noise," in *ICSLP*, 2000, vol. 1, pp. 770–773.

[5] B. Frey, L. Deng, A. Acero, and T. Kristjansson, "Algonquin: Iterating laplace's method to remove multiple types of acoustic distortion for robust speech recognition," in *EUROSPEECH*, 2001, pp. 901–904.

[6] J. S. Liu and R. Chen, "Sequential monte carlo methods for dynamic systems," *J. Am. Stat. Assoc*, vol. 93, pp. 1032–1044, 1998.

[7] W. K. Hastings, "Monte carlo sampling methods using markov chains and their applications," *Biometrika*, vol. 57, pp. 97–109, 1970.
